# Theoretical Analysis of Heuristic Search Methods for Online POMDPs

**Stéphane Ross**
McGill University
Montréal, Qc, Canada
sross12@cs.mcgill.ca

**Joelle Pineau**
McGill University
Montréal, Qc, Canada
jpineau@cs.mcgill.ca

**Brahim Chaib-draa**
Laval University
Québec, Qc, Canada
chaib@ift.ulaval.ca

## Abstract

Planning in partially observable environments remains a challenging problem, despite significant recent advances in offline approximation techniques. A few online methods have also been proposed recently, and proven to be remarkably scalable, but without the theoretical guarantees of their offline counterparts. Thus it seems natural to try to unify offline and online techniques, preserving the theoretical properties of the former, and exploiting the scalability of the latter. In this paper, we provide theoretical guarantees on an anytime algorithm for POMDPs which aims to reduce the error made by approximate offline value iteration algorithms through the use of an efficient online searching procedure. The algorithm uses search heuristics based on an error analysis of lookahead search, to guide the online search towards reachable beliefs with the most potential to reduce error. We provide a general theorem showing that these search heuristics are admissible, and lead to complete and $\epsilon$-optimal algorithms. This is, to the best of our knowledge, the strongest theoretical result available for online POMDP solution methods. We also provide empirical evidence showing that our approach is also practical, and can find (provably) near-optimal solutions in reasonable time.

## 1 Introduction

Partially Observable Markov Decision Processes (POMDPs) provide a powerful model for sequential decision making under state uncertainty. However exact solutions are intractable in most domains featuring more than a few dozen actions and observations. Significant efforts have been devoted to developing approximate offline algorithms for larger POMDPs [1, 2, 3, 4]. Most of these methods compute a policy over the entire belief space. This is both an advantage and a liability. On the one hand, it allows good generalization to unseen beliefs, and this has been key to solving relatively large domains. Yet it makes these methods impractical for problems where the state space is too large to enumerate. A number of compression techniques have been proposed, which handle large state spaces by projecting into a sub-dimensional representation [5, 6]. Alternately online methods are also available [7, 8, 9, 10, 11]. These achieve scalability by planning only at execution time, thus allowing the agent to only consider belief states that can be reached over some (small) finite planning horizon. However despite good empirical performance, both classes of approaches lack theoretical guarantees on the approximation. So it would seem we are constrained to either solving small to mid-size problems (near-)optimally, or solving large problems possibly badly.

This paper suggests otherwise, arguing that by combining offline and online techniques, we can preserve the theoretical properties of the former, while exploiting the scalability of the latter. In previous work [11], we introduced an anytime algorithm for POMDPs which aims to reduce the error made by approximate offline value iteration algorithms through the use of an efficient online searching procedure. The algorithm uses search heuristics based on an error analysis of lookahead search, to guide the online search towards reachable beliefs with the most potential to reduce error. In

this paper, we derive formally the heuristics from our error minimization point of view and provide theoretical results showing that these search heuristics are admissible, and lead to complete and $\epsilon$-optimal algorithms. This is, to the best of our knowledge, the strongest theoretical result available for online POMDP solution methods. Furthermore the approach works well with factored state representations, thus further enhancing scalability, as suggested by earlier work [2]. We also provide empirical evidence showing that our approach is computationally practical, and can find (provably) near-optimal solutions within a smaller overall time than previous online methods.

## 2  Background: POMDP

A POMDP is defined by a tuple $(S, A, \Omega, T, R, O, \gamma)$ where $S$ is the state space, $A$ is the action set, $\Omega$ is the observation set, $T : S \times A \times S \rightarrow [0, 1]$ is the state-to-state transition function, $R : S \times A \rightarrow \mathbb{R}$ is the reward function, $O : \Omega \times A \times S \rightarrow [0, 1]$ is the observation function, and $\gamma$ is the discount factor. In a POMDP, the agent often does not know the current state with full certainty, since observations provide only a partial indicator of state. To deal with this uncertainty, the agent maintains a belief state $b(s)$, which expresses the probability that the agent is in each state at a given time step. After each step, the belief state $b$ is updated using Bayes rule. We denote the belief update function $b' = \tau(b, a, o)$, defined as $b'(s') = \eta O(o, a, s') \sum_{s \in S} T(s, a, s')b(s)$, where $\eta$ is a normalization constant ensuring $\sum_{s \in S} b'(s) = 1$.

Solving a POMDP consists in finding an optimal policy, $\pi^* : \Delta S \rightarrow A$, which specifies the best action $a$ to do in every belief state $b$, that maximizes the expected *return* (i.e., expected sum of discounted rewards over the planning horizon) of the agent. We can find the optimal policy by computing the optimal value of a belief state over the planning horizon. For the infinite horizon, the optimal value function is defined as $V^*(b) = \max_{a \in A}[R(b, a) + \gamma \sum_{o \in \Omega} P(o|b, a)V^*(\tau(b, a, o))]$, where $R(b, a)$ represents the expected immediate reward of doing action $a$ in belief state $b$ and $P(o|b, a)$ is the probability of observing $o$ after doing action $a$ in belief state $b$. This probability can be computed according to $P(o|b, a) = \sum_{s' \in S} O(o, a, s') \sum_{s \in S} T(s, a, s')b(s)$. We also denote the value $Q^*(b, a)$ of a particular action $a$ in belief state $b$, as the return we will obtain if we perform $a$ in $b$ and then follow the optimal policy $Q^*(b, a) = R(b, a) + \gamma \sum_{o \in \Omega} P(o|b, a)V^*(\tau(b, a, o))$. Using this, we can define the optimal policy $\pi^*(b) = \operatorname{argmax}_{a \in A} Q^*(b, a)$.

While any POMDP problem has infinitely many belief states, it has been shown that the optimal value function of a finite-horizon POMDP is piecewise linear and convex. Thus we can define the optimal value function and policy of a finite-horizon POMDP using a finite set of $|S|$-dimensional hyper plans, called $\alpha$-vectors, over the belief state space. As a result, exact offline value iteration algorithms are able to compute $V^*$ in a finite amount of time, but the complexity can be very high. Most approximate *offline* value iteration algorithms achieve computational tractability by selecting a small subset of belief states, and keeping only those $\alpha$-vectors which are maximal at the selected belief states [1, 3, 4]. The precision of these algorithms depend on the number of belief points and their location in the space of beliefs.

## 3  Online Search in POMDPs

Contrary to offline approaches, which compute a complete policy determining an action for every belief state, an online algorithm takes as input the current belief state and returns the single action which is the best for this *particular belief state*. The advantage of such an approach is that it only needs to consider belief states that are reachable from the current belief state. This naturally provides a small set of beliefs, which could be exploited as in offline methods. But in addition, since online planning is done at every step (and thus generalization between beliefs is not required), it is sufficient to calculate only the *maximal value* for the current belief state, not the full optimal $\alpha$-vector. A lookahead search algorithm can compute this value in two simple steps.

First we build a tree of reachable belief states from the current belief state. The current belief is the top node in the tree. Subsequent belief states (as calculated by the $\tau(b, a, o)$ function) are represented using OR-nodes (at which we must choose an action) and actions are included in between each layer of belief nodes using AND-nodes (at which we must consider all possible observations). Note that in general the belief MDP could have a graph structure with cycles. Our algorithm simply handle

such structure by unrolling the graph into a tree. Hence, if we reach a belief that is already elsewhere in the tree, it will be duplicated.[1]

Second, we estimate the value of the current belief state by propagating value estimates up from the fringe nodes, to their ancestors, all the way to the root. An approximate value function is generally used at the fringe of the tree to approximate the infinite-horizon value. We are particularly interested in the case where a lower bound and an upper bound on the value of the fringe belief states is available, as this allows us to get a bound on the error at any specific node. The lower and upper bounds can be propagated to parent nodes according to:

$$U_T(b) = \begin{cases} U(b) & \text{if } b \text{ is a leaf in } T, \\ \max_{a \in A} U_T(b, a) & \text{otherwise}; \end{cases} \tag{1}$$

$$U_T(b, a) = R_B(b, a) + \gamma \sum_{o \in \Omega} P(o|b, a) U_T(\tau(b, a, o)); \tag{2}$$

$$L_T(b) = \begin{cases} L(b) & \text{if } b \text{ is a leaf in } T, \\ \max_{a \in A} L_T(b, a) & \text{otherwise}; \end{cases} \tag{3}$$

$$L_T(b, a) = R_B(b, a) + \gamma \sum_{o \in \Omega} P(o|b, a) L_T(\tau(b, a, o)); \tag{4}$$

where $U_T(b)$ and $L_T(b)$ represent the upper and lower bounds on $V^*(b)$ associated to belief state $b$ in the tree $T$, $U_T(b, a)$ and $L_T(b, a)$ represent corresponding bounds on $Q^*(b, a)$, and $L(b)$ and $U(b)$ are the bounds on fringe nodes, typically computed offline.

Performing a complete $k$-step lookahead search multiplies the error bound on the approximate value function used at the fringe by $\gamma^k$ ([13]), and thus ensures better value estimates. However, it has complexity exponential in $k$, and may explore belief states that have very small probabilities of occurring (and an equally small impact on the value function) as well as exploring suboptimal actions (which have no impact on the value function). We would evidently prefer to have a more efficient online algorithm, which can guarantee equivalent or better error bounds. In particular, we believe that the best way to achieve this is to have a search algorithm which uses estimates of error reduction as a criteria to guide the search over the reachable beliefs.

## 4  Anytime Error Minimization Search

In this section, we review the Anytime Error Minimization Search (AEMS) algorithm we had first introduced in [11] and present a novel mathematical derivation of the heuristics that we had suggested. We also provide new theoretical results describing sufficient conditions under which the heuristics are guaranteed to yield $\epsilon$-optimal solutions.

Our approach uses a best-first search of the belief reachability tree, where error minimization (at the root node) is used as the search criteria to select which fringe nodes to expand next. Thus we need a way to express the error on the current belief (i.e. root node) as a function of the error at the fringe nodes. This is provided in Theorem 1. Let us denote (i) $\mathcal{F}(T)$, the set of fringe nodes of a tree $T$; (ii) $e_T(b) = V^*(b) - L_T(b)$, the error function for node $b$ in the tree $T$; (iii) $e(b) = V^*(b) - L(b)$, the error at a fringe node $b \in \mathcal{F}(T)$; (iv) $h_T^{b_0, b}$, the unique action/observation sequence that leads from the root $b_0$ to belief $b$ in tree $T$; (v) $d(h)$, the depth of an action/observation sequence $h$ (number of actions); and (vi) $P(h|b_0, \pi^*) = \prod_{i=1}^{d(h)} P(h_o^i|b_0^{h_{i-1}}, h_a^i) \pi^*(b^{h_{i-1}}, h_a^i)$, the probability of executing the action/observation sequence $h$ if we follow the optimal policy $\pi^*$ from the root node $b_0$ (where $h_a^i$ and $h_o^i$ refers to the $i^{th}$ action and observation in the sequence $h$, and $b^{h_i}$ is the belief obtained after taking the $i$ first actions and observations from belief $b$. $\pi^*(b, a)$ is the probability that the optimal policy chooses action $a$ in belief $b$).

By abuse of notation, we will use $b$ to represent both a belief node in the tree and its associated belief[2].

**Theorem 1.** *In any tree $T$, $e_T(b_0) \leq \sum_{b \in \mathcal{F}(T)} \gamma^{d(h_T^{b_0,b})} P(h_T^{b_0,b}|b_0, \pi^*)e(b)$.*

*Proof.* Consider an arbitrary parent node $b$ in tree $T$ and let's denote $\hat{a}_b^T = \text{argmax}_{a \in A} L_T(b,a)$. We have $e_T(b) = V^*(b) - L_T(b)$. If $\hat{a}_b^T = \pi^*(b)$, then $e_T(b) = \gamma \sum_{o \in \Omega} P(o|b, \pi^*(b))e(\tau(b, \pi^*(b), o))$. On the other hand, when $\hat{a}_b^T \neq \pi^*(b)$, then we know that $L_T(b, \pi^*(b)) \leq L_T(b, \hat{a}_b^T)$ and therefore $e_T(b) \leq \gamma \sum_{o \in \Omega} P(o|b, \pi^*(b))e(\tau(b, \pi^*(b), o))$. Consequently, we have the following:

$$e_T(b) \leq \begin{cases} e(b) & \text{if } b \in \mathcal{F}(T) \\ \gamma \sum_{o \in \Omega} P(o|b, \pi^*(b))e_T(\tau(b, \pi^*(b), o)) & \text{otherwise} \end{cases}$$

Then $e_T(b_0) \leq \sum_{b \in \mathcal{F}(T)} \gamma^{d(h_T^{b_0,b})} P(h_T^{b_0,b}|b_0, \pi^*)e(b)$ can be easily shown by induction. $\qquad\square$

## 4.1 Search Heuristics

From Theorem 1, we see that the contribution of each fringe node to the error in $b_0$ is simply the term $\gamma^{d(h_T^{b_0,b})} P(h_T^{b_0,b}|b_0, \pi^*)e(b)$. Consequently, if we want to minimize $e_T(b_0)$ as quickly as possible, we should expand fringe nodes reached by the optimal policy $\pi^*$ that maximize the term $\gamma^{d(h_T^{b_0,b})} P(h_T^{b_0,b}|b_0, \pi^*)e(b)$ as they offer the greatest potential to reduce $e_T(b_0)$. This suggests us a sound heuristic to explore the tree in a best-first-search way. Unfortunately we do not know $V^*$ nor $\pi^*$, which are required to compute the terms $e(b)$ and $P(h_T^{b_0,b}|b_0, \pi^*)$; nevertheless, we can approximate them. First, the term $e(b)$ can be estimated by the difference between the lower and upper bound. We define $\hat{e}(b) = U(b) - L(b)$ as an estimate of the error introduced by our bounds at fringe node $b$. Clearly, $\hat{e}(b) \geq e(b)$ since $U(b) \geq V^*(b)$.

To approximate $P(h_T^{b_0,b}|b_0, \pi^*)$, we can view the term $\pi^*(b,a)$ as the probability that action $a$ is optimal in belief $b$. Thus, we consider an approximate policy $\hat{\pi}_T$ that represents the probability that action $a$ is optimal in belief state $b$ given the bounds $L_T(b,a)$ and $U_T(b,a)$ that we have on $Q^*(b,a)$ in tree $T$. More precisely, to compute $\hat{\pi}_T(b,a)$, we consider $Q^*(b,a)$ as a random variable and make some assumptions about its underlying probability distribution. Once cumulative distribution functions $F_T^{b,a}$, s.t. $F_T^{b,a}(x) = P(Q^*(b,a) \leq x)$, and their associated density functions $f_T^{b,a}$ are determined for each $(b,a)$ in tree $T$, we can compute the probability $\hat{\pi}_T(b,a) = P(Q^*(b,a') \leq Q^*(b,a) \forall a' \neq a) = \int_{-\infty}^{\infty} f_T^{b,a}(x) \prod_{a' \neq a} F_T^{b,a'}(x)dx$. Computing this integral may not be computationally efficient depending on how we define the functions $f_T^{b,a}$. We consider two approximations.

One possible approximation is to simply compute the probability that the Q-value of a given action is higher than its parent belief state value (instead of all actions' Q-value). In this case, we get $\hat{\pi}_T(b,a) = \int_{-\infty}^{\infty} f_T^{b,a}(x) F_T^b(x)dx$, where $F_T^b$ is the cumulative distribution function for $V^*(b)$, given bounds $L_T(b)$ and $U_T(b)$ in tree $T$. Hence by considering both $Q^*(b,a)$ and $V^*(b)$ as random variables with uniform distributions between their respective lower and upper bounds, we get:

$$\hat{\pi}_T(b,a) = \begin{cases} \eta \frac{(U_T(b,a)-L_T(b))^2}{U_T(b,a)-L_T(b,a)} & \text{if } U_T(b,a) > L_T(b), \\ 0 & \text{otherwise.} \end{cases} \tag{5}$$

where $\eta$ is a normalization constant such that $\sum_{a \in A} \hat{\pi}_T(b,a) = 1$. Notice that if the density function is 0 outside the interval between the lower and upper bound, then $\hat{\pi}_T(b,a) = 0$ for dominated actions, thus they are implicitly pruned from the search tree by this method.

A second practical approximation is:

$$\hat{\pi}_T(b,a) = \begin{cases} 1 & \text{if } a = \text{argmax}_{a' \in A} U_T(b,a'), \\ 0 & \text{otherwise.} \end{cases} \tag{6}$$

which simply selects the action that maximizes the upper bound. This restricts exploration of the search tree to those fringe nodes that are reached by sequence of actions that maximize the upper bound of their parent belief state, as done in the $AO^*$ algorithm [14]. The nice property of this approximation is that these fringe nodes are the only nodes that can potentially reduce the upper bound in $b_0$.

Using either of these two approximations for $\hat{\pi}_T$, we can estimate the error contribution $\hat{e}_T(b_0, b)$ of a fringe node $b$ on the value of root belief $b_0$ in tree $T$, as: $\hat{e}_T(b_0, b) = \gamma^{d(h_T^{b_0,b})} P(h_T^{b_0,b}|b_0, \hat{\pi}_T)\hat{e}(b)$. Using this as a heuristic, the next fringe node $\widetilde{b}(T)$ to expand in tree $T$ is defined as $\widetilde{b}(T) = \arg\max_{b \in \mathcal{F}(T)} \gamma^{d(h_T^{b_0,b})} P(h_T^{b_0,b}|b_0, \hat{\pi}_T)\hat{e}(b)$. We use **AEMS1**[3] to denote the heuristic that uses $\hat{\pi}_T$ as defined in Equation 5, and **AEMS2**[4] to denote the heuristic that uses $\hat{\pi}_T$ as defined in Equation 6.

## 4.2 Algorithm

Algorithm 1 presents the anytime error minimization search. Since the objective is to provide a near-optimal action within a finite allowed online planning time, the algorithm accepts two input parameters: $t$, the online search time allowed per action, and $\epsilon$, the desired precision on the value function.

---
**Algorithm 1** AEMS: Anytime Error Minimization Search
---
    **Function** SEARCH($t, \epsilon$)
    **Static :** $T$: an AND-OR tree representing the current search tree.
    $t_0 \leftarrow$ TIME()
    **while** TIME() $- t_0 \leq t$ **and not** SOLVED(ROOT($T$), $\epsilon$) **do**
        $b^* \leftarrow \widetilde{b}(T)$
        EXPAND($b^*$)
        UPDATEANCESTORS($b^*$)
    **end while**
    **return** $\arg\max_{a \in A} L_T(\text{ROOT}(T), a)$
---

The EXPAND function expands the tree one level under the node $b^*$ by adding the next action and belief nodes to the tree $T$ and computing their lower and upper bounds according to Equations 1-4. After a node is expanded, the UPDATEANCESTORS function simply recomputes the bounds of its ancestors according to Equations determining $b'(s')$, $V^*(b)$, $P(o|b, a)$ and $Q^*(b, a)$, as outlined in Section 2. It also recomputes the probabilities $\hat{\pi}_T(b, a)$ and the best actions for each ancestor node. To find quickly the node that maximizes the heuristic in the whole tree, each node in the tree contains a reference to the best node to expand in their subtree. These references are updated by the UPDATEANCESTORS function without adding more complexity, such that when this function terminates, we always know immediatly which node to expand next, as its reference is stored in the root node. The search terminates whenever there is no more time available, or we have found an $\epsilon$-optimal solution (verified by the SOLVED function). After an action is executed in the environment, the tree $T$ is updated such that our new current belief state becomes the root of $T$; all nodes under this new root can be reused at the next time step.

## 4.3 Completeness and Optimality

We now provide some sufficient conditions under which our heuristic search is guaranteed to converge to an $\epsilon$-optimal policy after a finite number of expansions. We show that the heuristics proposed in Section 4.1 satisfy those conditions, and therefore are *admissible*. Before we present the main theorems, we provide some useful preliminary lemmas.

**Lemma 1.** *In any tree $T$, the approximate error contribution $\hat{e}_T(b_0, b_d)$ of a belief node $b_d$ at depth $d$ is bounded by $\hat{e}_T(b_0, b_d) \leq \gamma^d \sup_b \hat{e}(b)$.*

*Proof.* $P(h_T^{b_0,b}|b_0, \hat{\pi}_T) \leq 1$ and $\hat{e}(b) \leq \sup_{b'} \hat{e}(b')$ for all $b$. Thus $\hat{e}_T(b_0, b_d) \leq \gamma^d \sup_b \hat{e}(b)$. $\qquad\square$

For the following lemma and theorem, we will denote $P(h_o|b_0, h_a) = \prod_{i=1}^{d(h)} P(h_o^i|b_0^{h_{i-1}}, h_a^i)$ the probability of observing the sequence of observations $h_o$ in some action/observation sequence $h$, given that the sequence of actions $h_a$ in $h$ is performed from current belief $b_0$, and $\widehat{\mathcal{F}}(T) \subseteq \mathcal{F}(T)$ the set of all fringe nodes in $T$ such that $P(h_T^{b_0,b}|b_0, \hat{\pi}_T) > 0$, for $\hat{\pi}_T$ defined as in Equation 6 (i.e.

the set of fringe nodes reached by a sequence of actions in which each action maximizes $U_T(b,a)$ in its respective belief state.)

**Lemma 2.** *For any tree $T$, $\epsilon > 0$, and $D$ such that $\gamma^D \sup_b \hat{e}(b) \leq \epsilon$, if for all $b \in \widehat{\mathcal{F}}(T)$, either $d(h_T^{b_0,b}) \geq D$ or there exists an ancestor $b'$ of $b$ such that $\hat{e}_T(b') \leq \epsilon$, then $\hat{e}_T(b_0) \leq \epsilon$.*

*Proof.* Let's denote $\hat{a}_b^T = \operatorname{argmax}_{a \in A} U_T(b,a)$. Notice that for any tree $T$, and parent belief $b \in T$, $\hat{e}_T(b) = U_T(b) - L_T(b) \leq U_T(b, \hat{a}_b^T) - L_T(b, \hat{a}_b^T) = \gamma \sum_{o \in \Omega} P(o|b, \hat{a}_b^T) \hat{e}_T(\tau(b, \hat{a}_b^T, o))$. Consequently, the following recurrence is an upper bound on $\hat{e}_T(b)$:

$$\hat{e}_T(b) \leq \begin{cases} \hat{e}(b) & \text{if } b \in \mathcal{F}(T) \\ \epsilon & \text{if } \hat{e}_T(b) \leq \epsilon \\ \gamma \sum_{o \in \Omega} P(o|b, \hat{a}_b^T) \hat{e}_T(\tau(b, \hat{a}_b^T, o)) & \text{otherwise} \end{cases}$$

By unfolding the recurrence for $b_0$, we get $\hat{e}_T(b_0) \leq \sum_{b \in A(T)} \gamma^{d(h_T^{b_0,b})} P(h_{T,o}^{b_0,b}|b_0, h_{T,a}^{b_0,b}) \hat{e}(b) + \epsilon \sum_{b \in B(T)} \gamma^{d(h_T^{b_0,b})} P(h_{T,o}^{b_0,b}|b_0, h_{T,a}^{b_0,b})$, where $B(T)$ is the set of parent nodes $b'$ having a descendant in $\widehat{\mathcal{F}}(T)$ such that $\hat{e}_T(b') \leq \epsilon$ and $A(T)$ is the set of fringe nodes $b''$ in $\widehat{\mathcal{F}}(T)$ not having an ancestor in $B(T)$. Hence if for all $b \in \widehat{\mathcal{F}}(T)$, $d(h_T^{b_0,b}) \geq D$ or there exists an ancestor $b'$ of $b$ such that $\hat{e}_T(b') \leq \epsilon$, then this means that for all $b$ in $A(T)$, $d(h_T^{b_0,b}) \geq D$, and therefore, $\hat{e}_T(b_0) \leq \gamma^D \sup_b \hat{e}(b) \sum_{b' \in A(T)} P(h_{T,o}^{b_0,b'}|b_0, h_{T,a}^{b_0,b'}) + \epsilon \sum_{b' \in B(T)} P(h_{T,o}^{b_0,b'}|b_0, h_{T,a}^{b_0,b'}) \leq \epsilon \sum_{b' \in A(T) \cup B(T)} P(h_{T,o}^{b_0,b'}|b_0, h_{T,a}^{b_0,b'}) = \epsilon$. $\qquad\square$

**Theorem 2.** *For any tree $T$ and $\epsilon > 0$, if $\hat{\pi}_T$ is defined such that $\inf_{b,T|\hat{e}_T(b) > \epsilon} \hat{\pi}_T(b, \hat{a}_b^T) > 0$ for $\hat{a}_b^T = \operatorname{argmax}_{a \in A} U_T(b,a)$, then Algorithm 1 using $\widetilde{b}(T)$ is complete and $\epsilon$-optimal.*

*Proof.* If $\gamma = 0$, then the proof is immediate. Consider now the case where $\gamma \in (0,1)$. Clearly, since $U$ is bounded above and $L$ is bounded below, then $\hat{e}$ is bounded above. Now using $\gamma \in (0,1)$, we can find a positive integer $D$ such that $\gamma^D \sup_b \hat{e}(b) \leq \epsilon$. Let's denote $\mathcal{A}_b^T$ the set of ancestor belief states of $b$ in the tree $T$, and given a finite set $A$ of belief nodes, let's define $\hat{e}_T^{min}(A) = \min_{b \in A} \hat{e}_T(b)$. Now let's define $\mathcal{T}_b = \{T | T \text{ finite}, b \in \widehat{\mathcal{F}}(T), \hat{e}_T^{min}(\mathcal{A}_b^T) > \epsilon\}$ and $\mathcal{B} = \{b | \hat{e}(b) \inf_{T \in \mathcal{T}_b} P(h_T^{b_0,b}|b_0, \hat{\pi}_T) > 0, d(h_T^{b_0,b}) \leq D\}$. Clearly, by the assumption that $\inf_{b,T|\hat{e}_T(b) > \epsilon} \hat{\pi}_T(b, \hat{a}_b^T) > 0$, then $\mathcal{B}$ contains all belief states $b$ within depth $D$ such that $\hat{e}(b) > 0$, $P(h_{T,o}^{b_0,b}|b_0, h_{T,a}^{b_0,b}) > 0$ and there exists a finite tree $T$ where $b \in \widehat{\mathcal{F}}(T)$ and all ancestors $b'$ of $b$ have $\hat{e}_T(b') > \epsilon$. Furthermore, $\mathcal{B}$ is finite since there are only finitely many belief states within depth $D$. Hence there exist a $E_{min} = \min_{b \in \mathcal{B}} \gamma^{d(h_T^{b_0,b})} \hat{e}(b) \inf_{T \in \mathcal{T}_b} P(h_T^{b_0,b}|b_0, \hat{\pi}_T)$. Clearly, $E_{min} > 0$ and we know that for any tree $T$, all beliefs $b$ in $\mathcal{B} \cap \widehat{\mathcal{F}}(T)$ have an approximate error contribution $\hat{e}_T(b_0, b) \geq E_{min}$. Since $E_{min} > 0$ and $\gamma \in (0,1)$, there exist a positive integer $D'$ such that $\gamma^{D'} \sup_b \hat{e}(b) < E_{min}$. Hence by Lemma 1, this means that Algorithm 1 cannot expand any node at depth $D'$ or beyond before expanding a tree $T$ where $\mathcal{B} \cap \widehat{\mathcal{F}}(T) = \emptyset$. Because there are only finitely many nodes within depth $D'$, then it is clear that Algorithm 1 will reach such tree $T$ after a finite number of expansions. Furthermore, for this tree $T$, since $\mathcal{B} \cap \widehat{\mathcal{F}}(T) = \emptyset$, we have that for all beliefs $b \in \widehat{\mathcal{F}}(T)$, either $d(h_T^{b_0,b}) \geq D$ or $\hat{e}_T^{min}(\mathcal{A}_b^T) \leq \epsilon$. Hence by Lemma 2, this implies that $\hat{e}_T(b_0) \leq \epsilon$, and consequently Algorithm 1 will terminate after a finite number of expansions (SOLVED$(b_0, \epsilon)$ will evaluate to true) with an $\epsilon$-optimal solution (since $e_T(b_0) \leq \hat{e}_T(b_0)$). $\qquad\square$

From this last theorem, we notice that we can potentially develop many different admissible heuristics for Algorithm 1; the main sufficient condition being that $\hat{\pi}_T(b,a) > 0$ for $a = \operatorname{argmax}_{a' \in A} U_T(b, a')$. It also follows from this theorem that the two heuristics described above, AEMS1 and AEMS2, are admissible. The following corollaries prove this:

**Corollary 1.** *Algorithm 1, using $\widetilde{b}(T)$, with $\hat{\pi}_T$ as defined in Equation 6 is complete and $\epsilon$-optimal.*

*Proof.* Immediate by Theorem 2 and the fact that $\hat{\pi}_T(b, \hat{a}_b^T) = 1$ for all $b, T$. $\qquad\square$

**Corollary 2.** *Algorithm 1, using $\widetilde{b}(T)$, with $\hat{\pi}_T$ as defined in Equation 5 is complete and $\epsilon$-optimal.*

*Proof.* We first notice that $(U_T(b,a) - L_T(b))^2/(U_T(b,a) - L_T(b,a)) \leq \hat{e}_T(b,a)$, since $L_T(b) \geq L_T(b,a)$ for all $a$. Furthermore, $\hat{e}_T(b,a) \leq \sup_{b'} \hat{e}(b')$. Therefore the normalization constant $\eta \geq (|A| \sup_b \hat{e}(b))^{-1}$. For $\hat{a}_b^T = \operatorname{argmax}_{a \in A} U_T(b,a)$, we have $U_T(b, \hat{a}_b^T) = U_T(b)$, and therefore $U_T(b, \hat{a}_b^T) - L_T(b) = \hat{e}_T(b)$. Hence this means that $\hat{\pi}_T(b, \hat{a}_b^T) = \eta(\hat{e}_T(b))^2/\hat{e}_T(b, \hat{a}_b^T) \geq$

$(|A|(\sup_{b'} \hat{e}(b'))^2)^{-1}(\hat{e}_T(b))^2$ for all $T$, $b$. Hence, for any $\epsilon > 0$, $\inf_{b,T|\hat{e}_T(b)>\epsilon} \hat{\pi}_T(b, \hat{a}_b^T) \geq (|A|(\sup_b \hat{e}(b))^2)^{-1}\epsilon^2 > 0$. Hence, corrolary follows from Theorem 2. $\square$

## 5  Experiments

In this section we present a brief experimental evaluation of Algorithm 1, showing that in addition to its useful theoretical properties, the empirical performance matches, and in some cases exceeds, that of other online approaches. The algorithm is evaluated in three large POMDP environments: Tag [1], RockSample [3] and FieldVisionRockSample (FVRS) [11]; all are implemented using a factored state representation. In each environments we compute the Blind policy[5] to get a lower bound and the FIB algorithm [15] to get an upper bound. We then compare performance of Algorithm 1 with both heuristics (AEMS1 and AEMS2) to the performance achieved by other online approaches (Satia [7], BI-POMDP [8], RTBSS [10]). For all approaches we impose a real-time constraint of 1 sec/action, and measure the following metrics: average return, average error bound reduction[6] (EBR), average lower bound improvement[7] (LBI), number of belief nodes explored at each time step, percentage of belief nodes reused in the next time step, and the average online time per action ($< 1$s means the algorithm found an $\epsilon$-optimal action)[8]. Satia, BI-POMDP, AEMS1 and AEMS2 were all implemented using the same algorithm since they differ only in their choice of search heuristic used to guide the search. RTBSS served as a base line for a complete $k$-step lookahead search using branch & bound pruning. All results were obtained on a Xeon 2.4 Ghz with 4Gb of RAM; but the processes were limited to use a max of 1Gb of RAM.

Table 1 shows the average value (over 1000+ runs) of the different statistics. As we can see from these results, AEMS2 provides the best average return, average error bound reduction and average lower bound improvement in all considered environments. The higher error bound reduction and lower bound improvement obtained by AEMS2 indicates that it can guarantee performance closer to the optimal. We can also observe that AEMS2 has the best average reuse percentage, which indicates that AEMS2 is able to guide the search toward the most probable nodes and allows it to generally maintain a higher number of belief nodes in the tree. Notice that AEMS1 did not perform very well, except in FVRS[5,7]. This could be explained by the fact that our assumption that the values of the actions are uniformly distributed between the lower and upper bounds is not valid in the considered environments.

Finally, we also examined how fast the lower and upper bounds converge if we let the algorithm run up to 1000 seconds on the initial belief state. This gives an indication of which heuristic would be the best if we extended online planning time past 1sec. Results for RockSample[7,8] are presented in Figure 2, showing that the bounds converge much more quickly for the AEMS2 heuristic.

## 6  Conclusion

In this paper we examined theoretical properties of online heuristic search algorithms for POMDPs. To this end, we described a general online search framework, and examined two admissible heuristics to guide the search. The first assumes that $Q^*(b, a)$ is distributed uniformly at random between the bounds (Heuristic AEMS1), the second favors an optimistic point of view, and assume the $Q^*(b, a)$ is equal to the upper bound (Heuristic AEMS2). We provided a general theorem that shows that AEMS1 and AEMS2 are admissible and lead to complete and $\epsilon$-optimal algorithms. Our experimental work supports the theoretical analysis, showing that AEMS2 is able to outperform online approaches. Yet it is equally interesting to note that AEMS1 did not perform nearly as well. This highlights the fact that not all admissible heuristics are equally useful. Thus it will be interesting in the future to develop further guidelines and theoretical results describing which subclasses of heuristics are most appropriate.

Figure 1: Comparison of different online search algorithm in different environments.

| Heuristic / Algorithm | Return ± 0.01 | EBR (%) ± 0.1 | LBI ± 0.01 | Belief Nodes - | Reuse (%) ±0.1 | Time (ms) ±1 |
|---|---|---|---|---|---|---|
| Tag ($|S| = 870, |A| = 5, |\Omega| = 30$) | | | | | | |
| RTBSS(5) | -10.30 | 22.3 | 3.03 | 45067 | 0 | 580 |
| Satia & Lave | -8.35 | 22.9 | 2.47 | 36908 | 10.0 | 856 |
| **AEMS1** | **-6.73** | **49.0** | **3.92** | **43693** | **25.1** | **814** |
| BI-POMDP | -6.22 | 76.2 | 7.81 | 79508 | 54.6 | 622 |
| **AEMS2** | **-6.19** | **76.3** | **7.81** | **80250** | **54.8** | **623** |
| RockSample[7,8] ($|S| = 12545, |A| = 13, |\Omega| = 2$) | | | | | | |
| Satia & Lave | 7.35 | 3.6 | 0 | 509 | 8.9 | 900 |
| **AEMS1** | **10.30** | **9.5** | **0.90** | **579** | **5.3** | **916** |
| RTBSS(2) | 10.30 | 9.7 | 1.00 | 439 | 0 | 896 |
| BI-POMDP | 18.43 | 33.3 | 4.33 | 2152 | 29.9 | 953 |
| **AEMS2** | **20.75** | **52.4** | **5.30** | **3145** | **36.4** | **859** |
| FVRS[5,7] ($|S| = 3201, |A| = 5, |\Omega| = 128$) | | | | | | |
| RTBSS(1) | 20.57 | 7.7 | 2.07 | 516 | 0 | 254 |
| BI-POMDP | 22.75 | 11.1 | 2.08 | 4457 | 0.4 | 923 |
| Satia & Lave | 22.79 | 11.1 | 2.05 | 3683 | 0.4 | 947 |
| **AEMS1** | **23.31** | **12.4** | **2.24** | **3856** | **1.4** | **942** |
| **AEMS2** | **23.39** | **13.3** | **2.35** | **4070** | **1.6** | **944** |

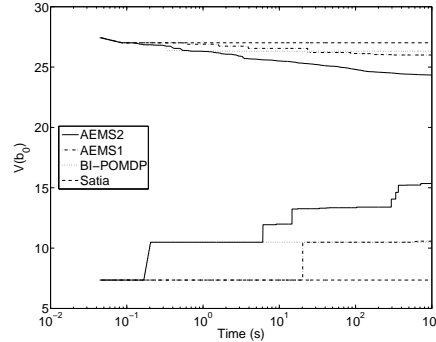

Figure 2: Evolution of the upper / lower bounds on the initial belief state in *RockSample[7,8]*.

## Acknowledgments

This research was supported by the Natural Sciences and Engineering Research Council of Canada (NSERC) and the Fonds Québécois de la Recherche sur la Nature et les Technologies (FQRNT).

## Footnotes

[1] We are considering using a technique proposed in the LAO* algorithm [12] to handle cycle, but we have not investigated this fully, especially in terms of how it affects the heuristic value presented below.

[2] e.g. $\sum_{b \in \mathcal{F}(T)}$ should be interpreted as a sum over all fringe nodes in the tree, while $e(b)$ to be the error associated to the belief in fringe node $b$.

[3]This heuristic is slightly different from the AEMS1 heuristic we had introduced in [11].

[4]This is the same as the AEMS2 heuristic we had introduced in [11].

[5]The policy obtained by taking the combination of the $|A|$ $\alpha$-vectors that each represents the value of a policy performing the same action in every belief state.

[6]The error bound reduction is defined as $1 - \frac{U_T(b_0)-L_T(b_0)}{U(b_0)-L(b_0)}$, when the search process terminates on $b_0$

[7]The lower bound improvement is defined as $L_T(b_0) - L(b_0)$, when the search process terminates on $b_0$

[8]For RTBSS, the maximum search depth under the 1sec time constraint is show in parenthesis.

## References

[1] J. Pineau. *Tractable planning under uncertainty: exploiting structure*. PhD thesis, Carnegie Mellon University, Pittsburgh, PA, 2004.

[2] P. Poupart. *Exploiting structure to efficiently solve large scale partially observable Markov decision processes*. PhD thesis, University of Toronto, 2005.

[3] T. Smith and R. Simmons. Point-based POMDP algorithms: improved analysis and implementation. In *UAI*, 2005.

[4] M. T. J. Spaan and N. Vlassis. Perseus: randomized point-based value iteration for POMDPs. *JAIR*, 24:195–220, 2005.

[5] N. Roy and G. Gordon. Exponential family PCA for belief compression in POMDPs. In *NIPS*, 2003.

[6] P. Poupart and C. Boutilier. Value-directed compression of POMDPs. In *NIPS*, 2003.

[7] J. K. Satia and R. E. Lave. Markovian decision processes with probabilistic observation of states. *Management Science*, 20(1):1–13, 1973.

[8] R. Washington. BI-POMDP: bounded, incremental partially observable Markov model planning. In *4th Eur. Conf. on Planning*, pages 440–451, 1997.

[9] D. McAllester and S. Singh. Approximate Planning for Factored POMDPs using Belief State Simplification. In *UAI*, 1999.

[10] S. Paquet, L. Tobin, and B. Chaib-draa. An online POMDP algorithm for complex multiagent environments. In *AAMAS*, 2005.

[11] S. Ross and B. Chaib-draa. AEMS: an anytime online search algorithm for approximate policy refinement in large POMDPs. In *IJCAI*, 2007.

[12] E. A. Hansen and S. Zilberstein. LAO * : A heuristic search algorithm that finds solutions with loops. *Artificial Intelligence*, 129(1-2):35–62, 2001.

[13] M. L. Puterman. *Markov Decision Processes: Discrete Stochastic Dynamic Programming*. John Wiley & Sons, Inc., New York, NY, USA, 1994.

[14] N.J. Nilsson. *Principles of Artificial Intelligence*. Tioga Publishing, 1980.

[15] M. Hauskrecht. Value-function approximations for POMDPs. *JAIR*, 13:33–94, 2000.
